# EMPATH:
# Face, Emotion, and Gender
# Recognition Using Holons

**Garrison W. Cottrell**
Computer Science and Engineering Dept.
Institute for Neural Computation
University of California San Diego
La Jolla, CA 92093

**Janet Metcalfe**
Department of Psychology
Dartmouth College
Hanover, NH 03755

## Abstract

The dimensionality of a set of 160 face images of 10 male and 10 female subjects is reduced from 4096 to 40 via an autoencoder network. The extracted features do not correspond to the features used in previous face recognition systems (Kanade, 1973), such as ratios of distances between facial elements. Rather, they are whole-face features we call *holons*. The holons are given to 1 and 2 layer back propagation networks that are trained to classify the input features for identity, feigned emotional state and gender. The automatically extracted holons provide a sufficient basis for all of the gender discriminations, 99% of the identity discriminations and several of the emotion discriminations among the training set. Network and human judgements of the emotions are compared, and it is found that the networks tend to confuse more distant emotions than humans do.

## 1 Introduction and motivation

We describe further research on the use of dimensionality-reduction networks for face recognition first described in (Cottrell & Fleming, 1990; Fleming & Cottrell, 1990). There, we demonstrated that an unsupervised autoencoding network was able to extract features from faces sufficient for identity discrimination. Here we extend that work to show that a network so trained can also recognize feigned emotional states.

Cottrell, Munro & Zipser (1987) showed that a back propagation network could be used for image compression. The network is trained to simply reproduce its input, and so can be seen as a non-linear version of Kohonen's (1977) auto-associator. However it must do this through a narrow channel of hidden units, so it must extract regularities from the input vectors during learning. Empirical analysis of the trained network showed that the hidden units span the principal subspace of the image vectors, with some noise on the first principal component due to network nonlinearity (Cottrell & Munro, 1988).

Although the network uses error-correction learning, no teacher other than the input is provided, so the learning can be regarded as *unsupervised*. We suggested that this network could be used for automatic feature extraction in a pattern recognition system. This is the approach taken here.

The model is shown in Figure 1. The image compression network extracts the features, and then the hidden unit representation so developed is given as input to one and two layer classification networks which yield identity, gender, and emotion as output. In previous work, we showed that the features developed by the model are holistic rather than discrete features, that they can combine to form faces that the model has never been trained on, and that they form a redintegrative memory, able to complete noisy or partial inputs (Kohonen et al., 1977). We have dubbed them *holons*.

## 2 Materials

The images that comprised the input to the network were selected from full face pictures taken of 10 females and 10 males. All subjects were introductory psychology students at the University of California, San Diego who received partial course credit for participating. Following the procedure outlined by Galton (1878), images were aligned along the axes of the eyes and mouth. These images were captured by a frame grabber, and reduced to 64×64 pixels by averaging. To prevent the use of first order statistics for discrimination, the images were normalized to have equal brightness and variance. The gray levels were linearly scaled to the range [0,.8]. Part of the training set and its reproduction by the autoencoder are shown in Figure 2.

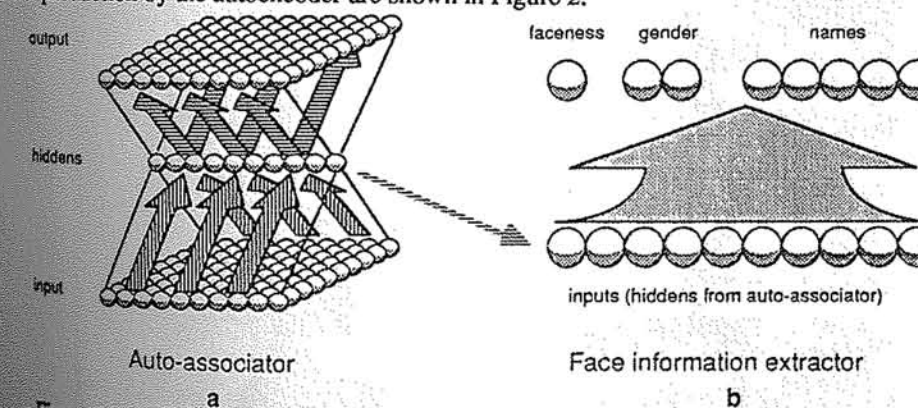

Auto-associator
a

Face information extractor
b

Figure 1:    The face recognition model. (a) An image compression network is trained first to compress the 4096 inputs into 40 hidden units. (b) The hidden units from the first network are used as inputs to various recognition networks.

Each column corresponds to one of 8 different feigned emotional expressions. Russell (1980) has shown that subjects' judgements of adjectives describing human emotions can be represented in a two-dimensional "emotion space" (Figure 3). The horizontal dimension can be characterized as pleasure/displeasure; the vertical dimension as high arousal/low arousal. Russell and his colleagues have shown using multidimensional scaling techniques (Russell, 1980, 1983; Russell & Bullock, 1986) that most common human emotions fall on a circle within this space. We chose adjectives from this circle to be the emotions that we asked our subjects to feign. The adjectives for each class are those in the numbered circles in Figure 3. If the subject did not respond well to one of the adjectives, others from the circled region were given as encouragement to form the proper facial expression. We labeled these classes with one adjective from each region: astonished, happy, pleased, relaxed, sleepy, bored, miserable and angry. The adjectives were presented in randomized order to offset possible carry over effects.

We found that subjects were enthusiastically expressive with certain of these emotions, such as astonished and delighted. However, despite claims of negative feelings when cued with adjectives such as miserable, bored and sleepy, the subjects did not overtly express these negative emotions very clearly.

## 3  Procedure

The whole image is input at once to our network, so the input layer is 64×64. We used 40 hidden units, and a 64×64 output layer, with a sigmoidal activation function with range [-1,1]. Due to the extreme difference in fan-in at the hidden and output layers (4096 vs. 40), differential learning rates were used at the two layers. Use of a single learning rate led to most of the hidden units becoming pinned at full off or on regardless

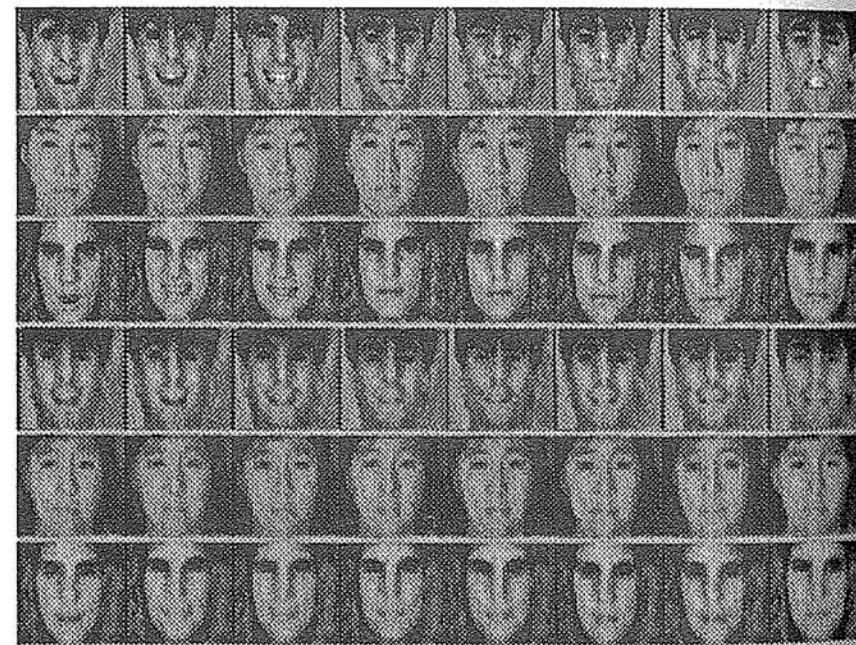

Figure 2:  Three subjects and their reproductions by the compression network.

of the input. We used a learning rate of .25 at the output layer during the first epoch, in order to quickly learn the bias, or "palette", then a rate of .1 was used for the remaining 49 epochs, where an epoch corresponds to the presentation of all 160 images. The hidden layer used a constant learning rate of .0001. The initial weight span was .1 (+/- .05). We used no momentum or weight decay. The average squared error per unit at the end was .0017. This corresponds to about 12 gray levels per pixel. Sample reproductions of trained images are shown in Figure 2.

The 40-element vectors produced by the hidden units of the compression network are then given as input to a single layer network that has a localist unit for every name and a unit for gender. A two-layer network with 20 hidden units is used for identifying which of the 8 emotion adjectives that were given to the subjects pertains to each image. The network is trained to produce .5 for the wrong answer, and 1 for the correct answer. The emotion network is trained for 1000 epochs, which reduces the total sum squared error to 22. To investigate how performance changed with further training, we trained this network for 2000 more epochs. 9 other networks were trained using the features from the same compression network for 1000 epochs from different initial random weights for comparison to human subjects on the same task.

## 4  Results

The criteria for correctness was that the output unit with the maximum activation must correspond to the correct answer. The network learned to discriminate 99% of the training set for identity. One image of one woman was taken for another. Sex discrimination was perfect. It was found that performance was better on these tasks without a hidden layer. The emotion classification network performed better with a

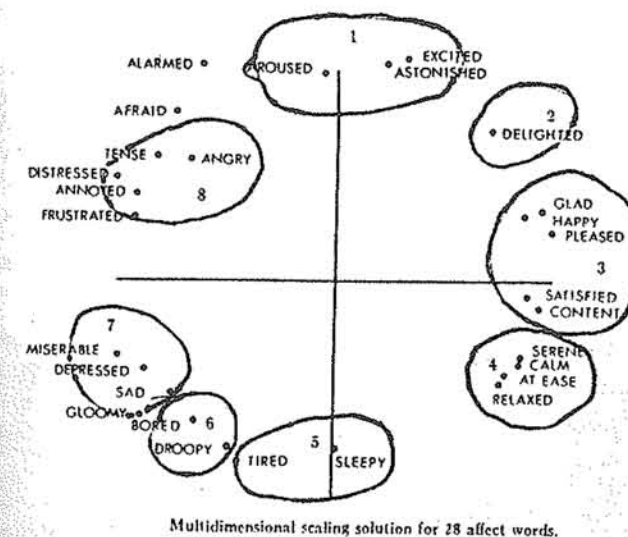

Multidimensional scaling solution for 28 affect words.

Figure 3:  The two-dimensional emotion space extracted via multi-dimensional scaling from similarity ratings. Data from Russell (1980).

Table 1: Percentage hits on each emotion (Generalization in parentheses)

| Emotion/Epochs | 1000 | | 2000 | | 3000 | |
|---|---|---|---|---|---|---|
| astonished | 100 | (40) | 100 | (60) | 100 | (40) |
| delighted | 55 | (20) | 75 | (40) | 90 | (40) |
| pleased | 80 | (100) | 90 | (40) | 80 | (40) |
| relaxed | 80 | (40) | 35 | (0) | 45 | (20) |
| sleepy | 20 | (0) | 70 | (0) | 80 | (0) |
| bored | 5 | (0) | 85 | (0) | 85 | (0) |
| miserable | 65 | (0) | 75 | (0) | 80 | (20) |
| angry | 85 | (40) | 85 | (0) | 70 | (0) |

hidden layer. However, the observation during data acquisition that negative emotions are poorly portrayed was confirmed by the network's early performance (See Table 1).

Initially, the network was much better at detecting positive states than negative ones. Later training improved some categories at the expense of others, suggesting that the network did not have enough capacity to perform all the discriminations. Generalization tests were performed on a small set of 5 subjects (40 faces in all), with the results shown in parentheses in Table 1. Generalization gets worse with more training, suggesting the network is becoming overtrained. The network is best at generalization on the positive emotions. This also suggests that the negative emotions are not easily discriminable in our data set. The generalization results, while not spectacular, should be considered in the light of the fact that the training set only contained 20 subjects, and it should be noted that the compression network was not trained on the images in this test set.

## 5   Internal representation

We investigated the representation formed by the compression network. We found the receptive fields of the hidden units in this network to be white noise. In order to extract the actual features used, we recorded the hidden unit activations as the network processed all 160 images. We formed the covariance matrix of the hidden unit activations, and extracted the principal components. Note that this operation "re-localizes" the principal components from the distributed representation used. The resulting vectors may be decompressed for viewing purposes. The results are shown in Figure 4.

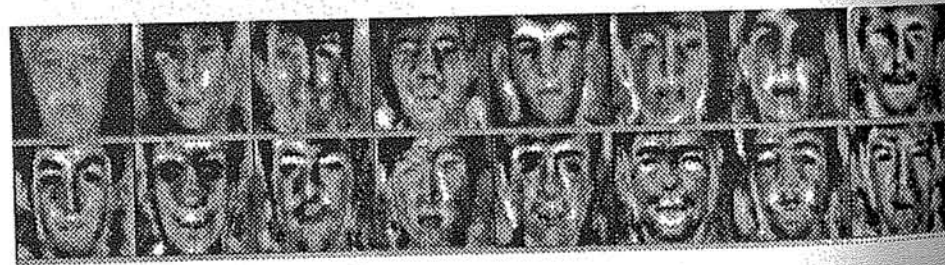

Figure 4:  Sixteen holons derived by PCA from hidden unit responses.

These are similar to the "eigenfaces" found by Turk & Pentland (submitted) in their principal components analysis of faces. Such a representation, if localized by a procedure such as Sanger's (1990), or as we have found in previous work develops at higher learning rates (Cottrell & Fleming, 1990), could provide a computational basis for the kinds of single cell recordings found in the STS of monkey cortex, without resorting to the notion of localist "grandmother" cells for each face.

## 6   Comparison with human subjects

In order to compare our network performance to that of human subjects on the same task, we tested human subjects on the same discriminations the networks were required to perform. 10 subjects were presented with one quarter of the training set (40 images of 5 subjects) 8 times in randomized order in each block (320 presentations total). On each presentation of an image, the subject was asked to make a Yes/No discrimination as to whether the image is a good example of one of the adjective class names.

Various factors (small sample size, large heterogeneity of variance) prevented a reliable statistical test of the model vs. the subjects. However, it is informative to compare the confusion matrices for the two pools of subjects (the other 9 network simulations are included here). All "yes" responses to each kind of face with each adjective are summed across subjects for the humans and the networks. The networks' responses were converted into "yes/no" responses by thresholding the outputs of the networks for each face, producing 8 yes/no responses. The threshold was chosen to produce approximately the same overall number of "yes" responses for the 10 networks as the 10 humans. The matrices are shown in Table 2. The rows correspond to the portrayals of the emotions, the columns are the adjectives presented with them. So, for example, across 10 subjects there were 45 instances of calling a "pleased" face a good example of "delighted" (out of a possible 50).

It is clear from the tables that there is a lot of regularity in the the human subjects data that is not totally captured by the model. The first three emotions/adjectives form a cluster, as do the last four. Since these adjectives are listed in the order of a tour around the circumference of Russell's circomplex model, the confusability of nearby emotions suggests that the clustering of descriptive adjectives is matched by a perceptual clustering of the facial expressions produced by a subject in response to those adjectives. However, rather than a diagonal band of confusability, as would be predicted by the circomplex, the positive/negative dimension appears to separate into two clusters. For example, anger and astonishment are separated more than would be expected from Russell's circomplex model (there is no "wrap-around" in this matrix). In between these two clusters, the adjective "relaxed" is seen by the subjects as compatible with nearly every facial category to some degree, but the "relaxed" faces are not compatible with the first three positive emotional categories.

The networks, while displaying some of the clustering shown in the human data, have higher entries along the diagonals, due to having been trained on this data, and more unusual confusions in regions where the human subjects (upper right and lower left) have practically no entries, such as "angry" labels on "delighted" and "pleased" faces. This may be due to forcing the networks to make as many responses as the humans. We found that a minor threshold change leads to many more responses, suggesting that we are over-extracting responses from the network.

**Table 2:** Confusion matrices for human and network subjects

| Face/Adj | Human | | | | | | | | Network | | | | | | | |
|---|---|---|---|---|---|---|---|---|---|---|---|---|---|---|---|---|
| | ast | del | ple | rel | sle | bor | mis | ang | ast | del | ple | rel | sle | bor | mis | ang |
| astonished | 37 | 24 | 21 | 9 | 0 | 1 | 0 | 0 | 50 | 8 | 5 | 14 | 13 | 9 | 4 | 9 |
| delighted | 6 | 29 | 32 | 24 | 4 | 6 | 1 | 0 | 10 | 44 | 27 | 7 | 8 | 6 | 7 | 20 |
| pleased | 2 | 45 | 48 | 18 | 0 | 0 | 0 | 0 | 1 | 29 | 50 | 1 | 3 | 2 | 10 | 23 |
| relaxed | 1 | 0 | 7 | 42 | 22 | 32 | 7 | 2 | 1 | 3 | 2 | 46 | 43 | 14 | 6 | 5 |
| sleepy | 0 | 0 | 0 | 28 | 31 | 33 | 19 | 1 | 13 | 4 | 2 | 39 | 49 | 2 | 10 | 4 |
| bored | 0 | 0 | 1 | 33 | 24 | 38 | 21 | 10 | 5 | 3 | 15 | 26 | 21 | 38 | 15 | 21 |
| miserable | 0 | 0 | 1 | 24 | 17 | 31 | 28 | 1 | 3 | 1 | 9 | 3 | 18 | 13 | 38 | 34 |
| angry | 4 | 0 | 0 | 16 | 11 | 12 | 13 | 3 | 3 | 4 | 14 | 3 | 2 | 11 | 25 | 49 |

## 7 Holons

This work demonstrates that, at least for our data set, dimensionality reduction is a useful preprocessing step that can maintain enough information for the recognition process. We term the representational units used by the compression network "holons". This is more than just another name for a distributed representation. By this we simply mean: Any representational element is a holon if its receptive field subtends the whole object being represented. Ideally we want to require that the information in a set of holons be maximally distributed: i.e., the average unit entropy is maximized. The latter restriction eliminates grandmother cells, insures that the representation be noise resistant, and also distributes the processing load evenly. A weak point of our definition is the difficulty of defining precisely the notion of a "whole object".

This definition applies to many distributed representational schemes, but does not apply to articulated ones such as the Wickelfeatures used by Rumelhart and McClelland (1987) in their past tense model as these only represent portions of the verb. On the other hand, we would not have holons for a "room", simply because we can not get a room to fill but not extend beyond our sensory surface at once. Given this meaning for the term, the units of area 17 are not holons, but the units in Superior Temporal Sulcus (STS) are. The main motivation for this definition is to give an alternative notion to the grandmother cell one for face cells in STS (Desimone et al., 1984).

## 8 Conclusions

We have shown that a network model that extracts features from its environment in an unsupervised manner can achieve near perfect recognition rates for identity discrimination and sex discrimination, even though the features were not extracted for that purpose. Where categories become "fuzzier", as in emotional states, the network's abilities are also limited. In particular, generalization to new faces is poor. In our preliminary study of human perception of these faces, we found support for the idea that when subjects are asked to produce expressions based on "near" adjectives in emotional space, they produce "near" expressions in perceptual space. These appear to fall in positive/negative clusters much more than the circomplex model would predict. However, this could be a fault of the subjects' abilities to portray the given emotions, rather than a fault of the circomplex model. Finally, we compared the networks' performance to that of humans. We found that the networks (when constrained to make as many responses as humans), while generally following the pattern of the human data,

produce several category confusions that humans do not.

## References

Cottrell, G. & Fleming, M. (1990). Face recognition using unsupervised feature extraction. In *Proceedings of the International Neural Network Conference*, Paris.

Cottrell, G, Munro, P. & Zipser, D. (1987). Learning internal representations of gray scale images: An example of extensional programming. In *Proc. Ninth Annual Cognitive Science Society Conference*, Seattle, Wa.

Cottrell, G.W. and Munro, P. (1988) Principal components analysis of images via back propagation. In *Proc. Soc. of Photo-Optical Instr. Eng.*, Cambridge, MA.

Desimone, R., Albright, T., Gross, C., and Bruce, C. (1984). Stimulus-selective properties of inferior temporal neurons in the Macaque. *J. Neuroscience, 4*, 2051-2062.

Fleming, M. & Cottrell, G. (1990). A neural network model of face recognition. In *Proceedings of the Int. Joint Conf. on Neural Networks*, San Diego, CA.

Galton, F. R. S. (1878). Composite Portraits. *Nature, 23*, 97-100.

Kanade, Takeo (1973). Picture processing system by computer complex and recognition of human faces. Unpublished Ph.D. Thesis, Dept. of Info. Science, Kyoto University.

Kohonen, T. Lehtio, P., Oja, E., Kortekangas, A., & Makisara, K. (1977). Demonstration of pattern processing properties of the optimal associative mappings. In *Proc Intl. Conf. on Cybernetics and Society*, Wash., D.C.

Russell, J. A. (1980). A circomplex model of affect. *Journal of Personality and Social Psychology, 39*, 1161-1178.

Russell, J. A. (1983). Pancultural aspects of the human conceptual organization of emotion. *Journal of Personality and Social Psychology, 45*, 1281-1288.

Russell, J. A. & Bullock, M. (1986). On the dimensions preschoolers use to interpret facial expressions of emotion. *Developmental Psychology, 22*, 97-102.

Rumelhart, D. E., Hinton, G. E., and Williams, R. J. (1986). Learning representations by back-propagating errors. *Nature, 323*, 533-536.

Rumelhart, D. & McClelland, J. On learning the past tenses of English verbs. In J.L. McClelland & D.E. Rumelhart (Eds.), *Parallel Distributed Processing, Vol 2.*, Cambridge, MA: MIT Press.

Sanger, T. D. (1989) Optimal unsupervised learning in a single-layer linear feedforward neural network. *Neural Networks*, 2, pp. 459-473.

Turk, M. & Pentland, A. (1990) Eigenfaces for recognition. (Submitted for publication).